# Unsupervised Learning of Human Motion Models

**Yang Song, Luis Goncalves, and Pietro Perona**
California Institute of Technology, 136-93, Pasadena, CA 9112 5, USA
{yangs,luis,perona}@vision.caltech.edu

## Abstract

This paper presents an unsupervised learning algorithm that can derive the probabilistic dependence structure of parts of an object (a moving human body in our examples) automatically from unlabeled data. The distinguished part of this work is that it is based on *unlabeled* data, i.e., the training features include both useful foreground parts and background clutter and the correspondence between the parts and detected features are unknown. We use decomposable triangulated graphs to depict the probabilistic independence of parts, but the unsupervised technique is not limited to this type of graph. In the new approach, labeling of the data (part assignments) is taken as hidden variables and the EM algorithm is applied. A greedy algorithm is developed to select parts and to search for the optimal structure based on the differential entropy of these variables. The success of our algorithm is demonstrated by applying it to generate models of human motion automatically from unlabeled real image sequences.

## 1 Introduction

Human motion detection and labeling is a very important but difficult problem in computer vision. Given a video sequence, we need to assign appropriate labels to the different regions of the image (*labeling*) and decide whether a person is in the image (*detection*). In [8, 7], a probabilistic approach was proposed by us to solve this problem. To detect and label a moving human body, a feature detector/tracker (such as corner detector) is first run to obtain the candidate features from a pair of frames. The combination of features is then selected based on maximum likelihood by using the joint probability density function formed by the position and motion of the body. Detection is performed by thresholding the likelihood. The lower part of Figure 1 depicts the procedure.

One key factor in the method is the probabilistic model of human motion. In order to avoid exponential combinatorial search, we use conditional independence property of body parts. In the previous work[8, 7], the independence structures were hand-crafted. In this paper, we focus on the the previously unresolved problem (upper part of Figure 1): how to learn the probabilistic independence structure of human motion automatically from *unlabeled* training data, meaning that the correspondence between the candidate features and the parts of the object is unknown. For example when we run a feature detector (such as Lucas-Tomasi-Kanade detector [10]) on real image sequences, the detected features can be from

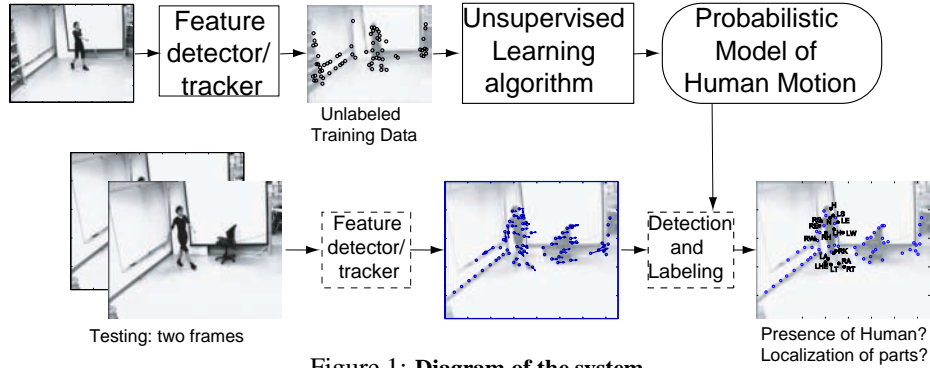

Figure 1: **Diagram of the system**.

target objects and background clutter with no identity attached to each feature. This case is interesting because the candidate features can be acquired automatically. Our algorithm leads to systems able to learn models of human motion completely automatically from real image sequences - unlabeled training features with clutter and occlusion.

We restrict our attention to triangulated models, since they both account for much correlation between the random variables that represent the position and motion of each body part, and they yield efficient algorithms. Our goal is to learn the best triangulated model, i.e., the one that reaches maximum likelihood with respect to the training data. Structure learning has been studied under the graphical model (Bayesian network) framework ([2, 4, 5, 6]). The distinguished part of this paper is that it is an unsupervised learning method based on *unlabeled* data, i.e., the training features include both useful foreground parts and background clutter and the correspondence between the parts and detected features are unknown. Although we work on triangulated models here, the unsupervised technique is not limited to this type of graph.

This paper is organized as follows. In section 2 we summarize the main facts about the triangulated probability model. In section 3 we address the learning problem when the training features are *labeled*, i.e., the parts of the model and the correspondence between the parts and observed features are known. In section 4 we address the learning problem when the training features are *unlabeled*. In section 5 we present some experimental results.

## 2 Decomposable triangulated graphs

Discovering the probability structure (conditional independence) among variables is important since it makes efficient learning and testing possible, hence some computationally intractable problems become tractable. Trees are good examples of modeling conditional (in)dependence [2, 6]. The decomposable triangulated graph is another type of graph which has been demonstrated to be useful for biological motion detection and labeling [8, 1].

A decomposable triangulated graph [1] is a collection of cliques of size three, where there is an elimination order of vertices such that when a vertex is deleted, it is only contained in one triangle and the remaining subgraph is again a collection of triangles until only one triangle left. Decomposable triangulated graphs are more powerful than trees since each node can be thought of as having two parents. Similarly to trees, efficient algorithms allow fast calculation of the maximum likelihood interpretation of a given set of data.

Conditional independence among random variables (parts) can be described by a decomposable triangulated graph. Let $\mathcal{S} = \{S1, S2, \ldots, SM\}$ be the set of $M$ parts, and $X_{Si}$, $1 \leq i \leq M$, is the measurement for $Si$. If the joint probability density function $P(X_{S1}, X_{S2}, \ldots, X_{SM})$ can be decomposed as a decomposable triangulated graph, it can

be written as,

$$P_{whole}(X_{S1}, X_{S2}, \ldots X_{SM})$$

$$= \prod_{t=1}^{T-1} P_{A_t | B_t C_t}(X_{A_t} | X_{B_t}, X_{C_t}) \cdot P_{A_T B_T C_T}(X_{A_T}, X_{B_T}, X_{C_T}) \qquad (1)$$

where $A_i, B_i, C_i \in \mathcal{S}$, $1 \leq i \leq T = M - 2$, $\{A_1, A_2, \ldots, A_T, B_T, C_T\} = \mathcal{S}$, and $(A_1, B_1, C_1), (A_2, B_2, C_2), \ldots, (A_T, B_T, C_T)$ are the cliques. $(A_1, A_2, \ldots, A_T)$ gives the elimination order for the decomposable graph.

## 3 Optimization of the decomposable triangulated graph

Suppose $\mathcal{X} = \{\overline{X}^1, \overline{X}^2, \ldots, \overline{X}^N\}$ are i.i.d samples from a probability density function, where $\overline{X}^n = (X_{S1}^n, \ldots, X_{SM}^n)$, $1 \leq n \leq N$, are labeled data. We want to find the decomposable triangulated graph $G$, such that $P(G|\mathcal{X})$ is maximized. $P(G|\mathcal{X})$ is the probability of graph $G$ being the 'correct' one given the observed data $\mathcal{X}$. Here we use $G$ to denote both the decomposable graph and the conditional (in)dependence depicted by the graph. By Bayes' rule, $P(G|\mathcal{X}) = P(\mathcal{X}|G)P(G)/P(\mathcal{X})$, therefore if we can assume the priors $P(G)$ are equal for different decompositions, then our goal is to find the structure $G$ which can maximize $P(\mathcal{X}|G)$. From the previous section, a decomposable triangulated graph $G$ is represented by $(A_1, B_1, C_1), (A_2, B_2, C_2), \ldots, (A_T, B_T, C_T)$, then $P(\mathcal{X}|G)$ can be computed as follows,

$$\log P(\mathcal{X}|G) \quad \cong \quad -N \cdot \sum_{t=1}^{T} h(X_{A_t} | X_{B_t}, X_{C_t}) - N \cdot h(X_{B_T}, X_{C_T}) \qquad (2)$$

where $h(\cdot)$ is differential entropy or conditional differential entropy [3] (we consider continuous random variables here). Equation (2) is an approximation which converges to equality for $N \to \infty$ due to the weak Law of Large numbers and definitions and properties of differential entropy [3, 2, 4, 5, 6]. We want to find the decomposition $(A_1, B_1, C_1), (A_2, B_2, C_2), \ldots, (A_T, B_T, C_T)$ such that the above equation can be maximized.

### 3.1 Greedy search

Though for tree cases, the optimal structure can be obtained efficiently by the maximum spanning tree algorithm [2, 6], for decomposable triangulated graphs, there is no existing algorithm which runs in polynomial time and guarantees to the optimal solution [9]. We develop a greedy algorithm to grow the graph by the property of decomposable graphs. For each possible choice of $C_T$ (the last vertex of the last triangle), find the best $B_T$ which can maximize $-h(X_{B_T}, X_{C_T})$, then get the best child of edge $(B_T, C_T)$ as $A_T$, i.e., the vertex (part) that can maximize $-h(X_{A_T} | X_{B_T}, X_{C_T})$. The next vertex is added one by one to the existing graph by choosing the best child of all the edges (legal parents) of the existing graph until all the vertices are added to the graph. For each choice of $C_T$, one such graph can be grown, so there are $M$ candidate graphs. The final result is the graph with the highest $\log P(\mathcal{X}|G)$ among the $M$ graphs.

The above algorithm is efficient. The total search cost is $M * (M - 1 + \sum_t ((2 * (T - t) + 1) * t))$, which is on the order of $M^4$. The algorithm is a greedy algorithm, with no guarantee that the global optimal solution could be found. Its effectiveness will be explored through experiments.

### 3.2 Computation of differential entropy - translation invariance

In the greedy search algorithm, we need to compute $h(X_{A_t}, X_{B_t}, X_{C_t})$ and $h(X_{B_t}, X_{C_t})$, $1 \leq t \leq T$. If we assume that they are jointly Gaussian, then the differential entropy can be computed by $\frac{1}{2} \log(2\pi e)^n |\Sigma|$, where $n$ is the dimension and $\Sigma$ is the covariance matrix.

In our applications, position and velocity are used as measurements for each body part, but humans can be present at different locations of the scene. In order to make the Gaussian assumption reasonable, translations need to be removed. Therefore, we use local coordinate system for each triangle $(A_t, B_t, C_t)$, i.e., we can take one body part (for example $A_t$) as the origin, and use relative positions for other body parts. More formally, let $\overline{x}$ denote a vector of positions $\overline{x} = (x_{A_t}, x_{B_t}, x_{C_t}, y_{A_t}, y_{B_t}, y_{C_t})^T$. Then if we describe positions relative to $A_t$, we obtain, $\overline{x}' = (x_{B_t} - x_{A_t}, x_{C_t} - x_{A_t}, y_{B_t} - y_{A_t}, y_{C_t} - y_{A_t})^T$. This can be written as $\overline{x}' = W\overline{x}$, where [12]

$$W = \begin{pmatrix} A & \underline{0} \\ \underline{0} & A \end{pmatrix}, \text{ with } A = \begin{pmatrix} -1 & 1 & 0 \\ -1 & 0 & 1 \end{pmatrix}.$$

In the greedy search algorithm, the differential entropy of all the possible triplets are needed and different triplets are with different origins. To reduce computational cost, notice that

$$\mu' = \frac{1}{N} \sum_{n=1}^{N} \overline{x}'^n = \frac{1}{N} \sum_{n=1}^{N} W\overline{x}^n = W \cdot \frac{1}{N} \sum_{n=1}^{N} \overline{x}^n = W\mu \qquad (4)$$

and

$$\Sigma' = W\Sigma W^T \qquad (5)$$

From the above equations, we can first estimate the mean $\mu$ and covariance $\Sigma$ of $\overline{X}^n$ (including all the body parts and without removing translation), then take the dimensions corresponding to the triangle and use equations (4) and (5) to get the mean and covariance for $(X_{A_t}, X_{B_t}, X_{C_t})$. Similar procedure can be applied to pairs (for example, $B_t$ can be taken as origin for $(B_t, C_t)$) to achieve translation invariant.

## 4 Unsupervised learning of the decomposable graph

In this section, we consider the case when only unlabeled data are available. Assume we have a data set of $N$ samples $\mathcal{X} = \{\overline{X}^1, \overline{X}^2, \ldots, \overline{X}^N\}$. Each sample $\overline{X}^n$, $1 \leq n \leq N$, is a group of detected features which contains the target object, but $\overline{X}^n$ is unlabeled, which means the correspondence between the candidate features and the parts of the object is unknown. For example when we run a feature detector (such as Lucas-Tomasi-Kanade detector [10]) on real image sequences, the detected features can be from target objects and background clutter with no identity attached to each feature. We want to select the useful composite parts of the object and learn the probability structure from $\mathcal{X}$.

### 4.1 All foreground parts observed

Here we first assume that all the foreground parts are observed for each sample. If the labeling for each $\overline{X}^n$ is taken as a hidden variable, then the EM algorithm can be used to learn the probability structure and parameters. Our method was developed from [11], but here we learn the probabilistic independence structure and all the candidate features are with the same type. Let $h_n$ denote the labeling for $\overline{X}^n$. If $\overline{X}^n$ contains $n_k$ features, then $h_n$ is an $n_k$-dimensional vector with each element taken a value from $\mathcal{S} \cup \{BG\}$ ($BG$ is the background clutter label). The observations for the EM algorithm are $\mathcal{X} = \{\overline{X}^1, \overline{X}^2, \ldots, \overline{X}^N\}$, the hidden variables are $\mathcal{H} = \{h_n\}_{n=1}^N$, and the parameters to optimize are the probability (in)dependence structure (i.e. the decomposable triangulated graph) and parameters for its associated probability density function. We use $G$ to represent both the probability structure and the parameters. If we assume that $\overline{X}^n$s are independent from each other and $h_n$ only depends on $\overline{X}^n$, then the likelihood function to maximize is,

$$\begin{aligned} L &= \log P(\mathcal{X}, G) = \log P(\mathcal{X}|G) + \log P(G) \\ &= \sum_{n=1}^{N} \log \sum_{h_{ni} \in H_n} P(\overline{X}^n, h_n = h_{ni}|G) + \log P(G) \end{aligned} \qquad (6)$$

where $h_{ni}$ is the $i$th possible labeling for $\overline{X}^n$, and $H_n$ is the set of all such labelings. Optimization directly over equation (6) is hard, and the EM algorithm solves the optimization problem iteratively. In EM, for each iteration $t$, we will optimize the function,

$$
\begin{aligned}
Q(G_t|G_{t-1}) &= E[\log P(\mathcal{X}, \mathcal{H}, G_t)|\mathcal{X}, G_{t-1}] \\
&= \sum_{n=1}^{N} E[\log P(\overline{X}^n, h_n, G_t)|\overline{X}^n, G_{t-1}] \\
&= \sum_{n=1}^{N} \sum_{h_{ni} \in H_n} P(h_n = h_{ni}|\overline{X}^n, G_{t-1}) \cdot \log P(\overline{X}^n, h_n = h_{ni}, G_t) \\
&= \sum_{n=1}^{N} \sum_{h_{ni} \in H_n} R_{ni} \log P(\overline{X}^n, h_n = h_{ni}, G_t) \qquad (7)
\end{aligned}
$$

where $R_{ni}$ is the probability of $h_n = h_{ni}$ given the observation $\overline{X}^n$ and the decomposable probability structure $G_{t-1}$. For each iteration $t$, $R_{ni}$ is a fixed number for a hypothesis $h_{ni}$. $R_{ni}$ can be computed as,

$$
R_{ni} = P(h_{ni}|\overline{X}^n, G_{t-1}) = P(h_{ni}, \overline{X}^n, G_{t-1})/\sum_{h_{ni}} P(h_{ni}, \overline{X}^n, G_{t-1}) \qquad (8)
$$

We will discuss the computation of $P(h_{ni}, \overline{X}^n, G_{t-1})$ below. Under the labeling hypothesis $h_n = h_{ni}$, $\overline{X}^n$ is divided into the foreground features $\overline{X}^n_{fg}$, which are parts of the object, and background (clutter) $\overline{X}^n_{bg}$. If the foreground features $\overline{X}^n_{fg}$ are independent of clutter $\overline{X}^n_{bg}$, then,

$$
\begin{aligned}
P(h_{ni}, \overline{X}^n, G) &= P(\overline{X}^n|h_{ni}, G)P(h_{ni}, G) \\
&= P(\overline{X}^n_{fg}|h_{ni}, G)P(\overline{X}^n_{bg}|h_{ni}, G)P(h_{ni}|G)P(G) \qquad (9)
\end{aligned}
$$

For simplicity, we will assume the priors $P(h_{ni}|G)$ are the same for different $h_{ni}$, and $P(G)$ are the same for different graph structures. If we assume uniform background densities [11, 8], then $P(\overline{X}^n_{bg}|h_{ni}, G) = (\frac{1}{A})^{n_k - M}$, where $A$ is the volume of the space a background feature lies in, is the same for different $h_{ni}$. Under probability decomposition $G$, $P(\overline{X}^n_{fg}|h_{ni}, G)$ can be computed as in equation (1). Therefore the maximization of equation (7) is equivalent to maximizing,

$$
\begin{aligned}
Q(G_t|G_{t-1}) &\sim \sum_{n=1}^{N} \sum_{h_{ni}} R_{ni} \log[P(\overline{X}^n_{fg}|h_{ni}, G_t)] \\
&= \sum_{n=1}^{N} \sum_{h_{ni}} R_{ni}[\sum_{t=1}^{T} \log P(X^{ni}_{A_t}|X^{ni}_{B_t}, X^{ni}_{C_t}) + \log P(X^{ni}_{B_T}, X^{ni}_{C_T})] \quad (10)
\end{aligned}
$$

For most problems, the number of possible labelings is very large (on the order of $n_k^M$), so it is computationally prohibitive to sum over all the possible $h_{ni}$ as in equation (10). However, if there is one hypothesis labeling $h_{ni}^*$ that is much better than other hypotheses,, i.e. $R_{ni}^*$ corresponding to $h_{ni}^*$ is much larger than other $R_{ni}$'s, then $R_{ni}^*$ can be taken as 1 and other $R_{ni}$'s as 0. Hence equation (10) can be approximated as,

$$
Q(G_t|G_{t-1}) \sim \sum_{n=1}^{N}[\sum_{t=1}^{T} \log P(X^{ni*}_{A_t}|X^{ni*}_{B_t}, X^{ni*}_{C_t}) + \log P(X^{ni*}_{B_T}, X^{ni*}_{C_T})] \qquad (11)
$$

where $X^{ni*}_{A_t}$, $X^{ni*}_{B_t}$ and $X^{ni*}_{C_t}$ are measurements corresponding to the best labeling $h_{ni}^*$. Comparing with equation (2) and also by the weak law of large numbers, we know for iteration $t$, if the best hypothesis $h_{ni}^*$ is used as the 'true' labeling, then the decomposable triangulated graph structure $G_t$ can be obtained through the algorithm described in section 3. One approximation we make here is that the best hypothesis labeling $h_{ni}^*$ for each $\overline{X}^n$ is really dominant among all the possible labelings so that hard assignment for labelings can be used. This is similar to the situation of K-means vs. mixture of Gaussian for clustering problems. We evaluate this approximation in experiments.

The whole algorithm can be summarized as follows. Given some random initial guess of the decomposable graph structure $G_0$ and its parameters, then for iteration $t$, ($t$ is from 1 until the algorithm converges),

E step: for each $\overline{X}^n$, use $G_{t-1}$ to find the best labeling $h_{ni}^*$ and then compute the differential entropies;

M step: use the differential entropies to run the greedy graph growing algorithm described in section 3 and get $G_t$.

### 4.2 Dealing with missing parts (occlusion)

So far we assume that all the parts are observed. In the case of some parts missing, the measurements for the missing parts can be taken as additional hidden variables [11], and the EM algorithm can be modified to handle the missing parts.

For each hypothesis $h_n$, let $X_o^n$ denote the measurements of the observed parts, $X_m^n$ be the measurements for the missing parts, and $X_{fg}^n = [X_o^{nT} X_m^{nT}]^T$ be the measurements of the whole object (to reduce clutter in the notation, we assume that the dimensions can be sorted in this way). For each EM iteration, we need to compute $\mu_{new}$ and $\Sigma_{new}$ to obtain the differential entropies and then $G_t$ with its parameters. Taking $h_n$ and $X_m^n$ as hidden variables, we can get,

$$\mu_{new} = \frac{1}{N} \sum_n E(X_{fg}^n) \tag{12}$$

$$\Sigma_{new} = \frac{1}{N} \sum_n E(X_{fg}^n - \mu_{new})(X_{fg}^n - \mu_{new})^T = \frac{1}{N} \sum_n E(X_{fg}^n X_{fg}^{nT}) - \mu_{new}\mu_{new}^T \tag{13}$$

Where $E(X_{fg}^n) = [X_o^{n*T} \ E(X_m^{nT})]^T$, and $E(X_{fg}^n X_{fg}^{nT}) = \begin{bmatrix} X_o^{n*} X_o^{n*T} & X_o^{n*} E(X_m^{nT}) \\ E(X_m^n) X_o^{n*T} & E(X_m^n X_m^{nT}) \end{bmatrix}$.

All the expectations $E(\cdot)$ are conditional expectations with respect to $\overline{X}^n$, $h_n = h_{ni}^*$ and decomposable graph structure $G_{t-1}$. Therefore, $X_o^{n*}$ are the measurements of the observed foreground parts under $h_n = h_{ni}^*$. Since $G_{t-1}$ is Gaussian distributed, conditional expectation $E(X_m^n)$ and $E(X_m^n X_m^{nT})$ can be computed from observed parts $X_o^{n*}$ and the mean and covariance matrix of $G_{t-1}$.

## 5 Experiments

We tested the greedy algorithm on labeled motion capture data (Johansson displays) as in [8], and the EM-like algorithm on unlabeled detected features from real image sequences.

### 5.1 Motion capture data

Our motion capture data consist of the 3-D positions of 14 markers fixed rigidly on a subject's body. These positions were tracked with 1mm accuracy as the subject walked back and forth, and projected to 2-D.

Under Gaussian assumption, we first estimated the joint probability density function (mean and covariance) of the data. From the estimated mean and covariance, we can compute differential entropies for all the possible triplets and pairs and further run the greedy search algorithm to find the approximated best triangulated model. Figure 2(a) shows the expected likelihood (differential entropy) of the estimated joint pdf, of the best triangulated model from the greedy algorithm, of the hand-constructed model from [8], and of randomly generated models. The greedy model is clearly superior to the hand-constructed model and the random models. The gap to the original joint pdf is partly due to the strong conditional independence assumptions of the triangulated model, which are an approximation of the true data's pdf. Figure 2(b) shows the expected likelihood using 50 synthetic datasets. Since these datasets were generated from 50 triangulated models, the greedy algorithm (solid

curve) can match the true model (dashed curve) extremely well. The solid line with error bars are the expected likelihoods of random triangulated models.

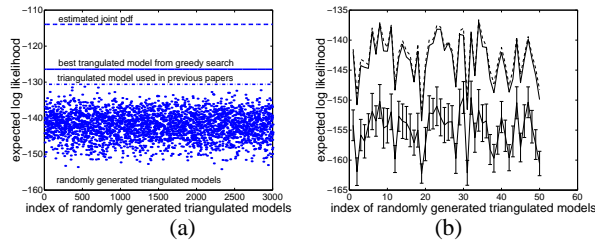

(a)                    (b)

Figure 2: **Evaluation of greedy search**.

## 5.2 Real image sequences

There are three types of sequences used here: (I) a subject walks from left to right (Figure 3(a,b)); (II) a subject walks from right to left; (III) a subject rides a bike from left to right (Figure3(c,d)). Left-to-right walking motion models were learned from type I sequences and tested on all types of sequences to see if the learned model can detect left-to-right walking and label the body parts correctly. The candidate features were obtained from a Lucas-Tomasi-Kanade algorithm [10] on two frames. We used two frames to simulate the difficult situation, where due to extreme body motion or to loose and textured clothing and occlusion, tracking is extremely unreliable and each feature's lifetime is short.

**Evaluation of the EM-like algorithm.** As described in section 4.1, one approximation we made is taking the best hypothesis labeling instead of summing over all the possible hypotheses (equation (11)). This approximation was evaluated by checking how the log-likelihoods evolve with EM iterations and if they converge. Figure 4(a) shows the results of learning a 12-feature model. We used random initializations, and each curve of Figure 4(a) corresponds to one such random initialization. From Figure 4(a) we can see that generally the log-likelihoods grow and converge well with the iterations of EM.

**Models obtained.** Figure 4 (b) and (c) show the best model obtained after we ran the EM algorithms for 11 times. Figure 4(b) gives the mean positions and mean velocities (shown in arrows) of the parts. Figure 4(c) shows the learned decomposable triangulated probabilistic structure. The letter labels show the body parts correspondence.

Figure 3 shows samples of the results. The red dots (with letter labels) are the maximum likelihood configuration from the left-to-right walking model. The horizontal bar at the bottom left of each frame shows the likelihood of the best configuration. The short vertical bar gives the threshold where $P_{falsealarm} = 1 - P_{detection}$ for all the test data. If

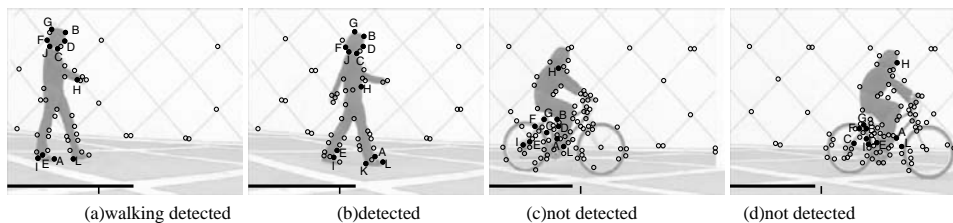

(a)walking detected        (b)detected        (c)not detected        (d)not detected

Figure 3: **Sample frames** from left-to-right walking (a-b) and biking sequences (c-d). The dots (either filled or empty) are the features selected by Tomasi-Kanade algorithm [10] on two frames. The filled dots (with letter labels) are the maximum likelihood configuration from the left-to-right walking model. The horizontal bar at the bottom left of each frame shows the likelihood of the best configuration. The short vertical bar gives the threshold for detection.

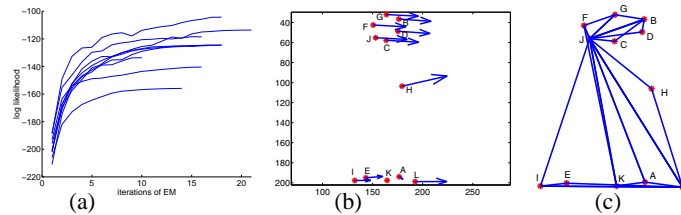

Figure 4: (a) Evaluation of the EM-like algorithm. Log-likelihood vs. iterations of EM for different random initializations. (b) and (c) show the best model obtained after we ran the EM-like algorithms for 11 times.

the likelihood is greater than the threshold, a left-to-right walking person is detected.The detection rate is 100% for the left-to-right walking vs. right-to-left walking, and 87% for the left-to-right walking vs. left-to-right biking.

## 6    Conclusions

In this paper we have described a method for learning the structure and parameters of a decomposable triangulated graph in an unsupervised fashion from unlabeled data. We have applied this method to learn models of biological motion that can be used to reliably detect and label biological motion.

## Acknowledgments

Funded by the NSF Engineering Research Center for Neuromorphic Systems Engineering (CNSE) at Caltech (NSF9402726), and by an NSF National Young Investigator Award to PP (NSF9457618). We thank Charless Fowlkes for bringing the Chow and Liu's paper to our attention. We thank Xiaolin Feng for providing the real image sequences.

## References

[1]  Y. Amit and A. Kong,  "Graphical templates for model registration",  *IEEE Transactions on Pattern Analysis and Machine Intelligence*, 18:225–236, 1996.

[2]  C.K. Chow and C.N. Liu,  "Approximating discrete probability distributions with dependence trees", *IEEE Transactions on Information Theory*, 14:462–467, 1968.

[3]  T.M. Cover and J.A. Thomas, *Elements of Information Theory*, John Wiley and Sons, 1991.

[4]  N. Friedman and M. Goldszmidt,  "Learning bayesian networks from data",  Technical report, AAAI 1998 Tutorial, http://robotics.stanford.edu/people/nir/tutorial/, 1998.

[5]  M.I. Jordan, editor, *Learning in Graphical Models*, MIT Press, 1999.

[6]  M. Meila and M.I. Jordan,  "Learning with mixtures of trees",  *Journal of Machine Learning Rearch*, 1:1–48, 2000.

[7]  Y. Song, X. Feng, and P. Perona,  "Towards detection of human motion",  In *Proc. IEEE CVPR 2000*, volume 1, pages 810–817, June 2000.

[8]  Y. Song, L. Goncalves, E. Di Bernardo, and P. Perona,  "Monocular perception of biological motion in johansson displays", *Computer Vision and Image Understanding*, 81:303–327, 2001.

[9]  Nathan Srebro,  "Maximum likelihood bounded tree-width markov networks",  In *UAI*, pages 504–511, San Francisco, CA, 2001.

[10]  C. Tomasi and T. Kanade,  "Detection and tracking of point features", *Tech. Rep. CMU-CS-91-132,Carnegie Mellon University*, 1991.

[11]  M. Weber, M. Welling, and P. Perona,  "Unsupervised learning of models for recognition",  In *Proc. ECCV*, volume 1, pages 18–32, June/July 2000.

[12]  Markus Weber, *Unsupervised Learning of Models for Object Recognition*, Ph.d. thesis, Caltech, May 2000.
